# On the Convergence of Leveraging[*]

**Gunnar Rätsch**[†]**, Sebastian Mika**[‡] **and Manfred K. Warmuth**[§]

[†]RSISE, Australian National University, Canberra, ACT 0200 Australia
[‡]Fraunhofer FIRST, Kekuléstr. 7, 12489 Berlin, Germany
[§]University of California at Santa Cruz, CA 95060, USA
*raetsch@csl.anu.edu.au, mika@first.fhg.de, manfred@cse.ucsc.edu*

## Abstract

We give an unified convergence analysis of ensemble learning methods including e.g. AdaBoost, Logistic Regression and the Least-Square-Boost algorithm for regression. These methods have in common that they iteratively call a base learning algorithm which returns hypotheses that are then linearly combined. We show that these methods are related to the *Gauss-Southwell method* known from numerical optimization and state *non-asymptotical* convergence results for all these methods. Our analysis includes $\ell_1$-norm regularized cost functions leading to a clean and general way to regularize ensemble learning.

## 1 Introduction

We show convergence rates of ensemble learning methods such as AdaBoost [10], Logistic Regression (LR) [11, 5] and the Least-Square (LS) regression algorithm called LS-Boost [12]. These algorithms have in common that they iteratively call a base learning algorithm $L$ (also called *weak learner*) on a weighted training sample. The base learner is expected to return in each iteration $t$ a hypothesis $\hat{h}_t$ from some hypothesis set of weak hypotheses $\mathcal{H}$ that has small *weighted training error*. This is the weighted number of false predictions in classification and weighted estimation error in regression. These hypotheses are then linearly *combined* to form the final hypothesis $f_{\hat{\alpha}}(\mathbf{x}) = \sum_t \hat{\alpha}_t \hat{h}_t(\mathbf{x})$; in classification one uses the sign of $f_{\hat{\alpha}}$ for prediction. The hypothesis coefficient $\hat{\alpha}_t$ is determined at iteration $t$, such that a certain objective is minimized or approximately minimized, and is fixed for later iterations. Here we will work out sufficient conditions on the base learning algorithm to achieve *linear convergence* to the minimum of an associated loss function $G$. This means that for any starting condition the minimum can be reached with precision $\epsilon > 0$ in only $\mathcal{O}(\log(1/\epsilon))$ iterations.

**Relation to Previous Work** In the original work on AdaBoost it has been shown that the optimization objective (which is an upper bound on the training error) converges exponentially fast to zero, if the base learner is consistently better than random guessing, i.e. its weighted training error $\epsilon$ is always smaller than some constant $\gamma$ with $\gamma < \frac{1}{2}$. In this case the convergence is known to be linear (i.e. exponentially decreasing) [10]. One can easily show that this is the case when the data is separable:[1] If the data is not separable, the

---

[*]Supported by DFG grants MU 987/1-1, JA 379/9-1 and NSF grant CCR 9821087; we gratefully acknowledge help from B. Borchers, P. Spellucci, R. Israel and S. Lemm. This work has been done, while G. Rätsch was at Fraunhofer FIRST, Berlin.

[1]We call the data separable, if there exists $\boldsymbol{\alpha}$ such that $f_{\boldsymbol{\alpha}}(\mathbf{x})$ separates the training examples.

weighted training error $\epsilon$ cannot be upper bounded by a constant smaller $\frac{1}{2}$, otherwise one could use AdaBoost to find a separation using the aforementioned convergence result.[2]

For AdaBoost and Logistic Regression it has been shown [5] that they generate a combined hypothesis *asymptotically* minimizing a loss functional G only depending on the output of the combined hypothesis $f_{\alpha}$. This holds for the non-separable case; however, the assumed conditions in [5] on the performance of the base learner are rather strict and can usually not be satisfied in practice. Although the analysis in [5] holds in principle for any strictly convex cost function of Legendre-type (e.g. [24], p. 258, and [1]), one needs to show the existence of a so-called *auxiliary function* [7, 5] for each cost function other than the exponential or the logistic loss. This can indeed be done [cf. 19, Section 4.2], but in any case only leads to asymptotic results. In the present work we can also show rates of convergence.

In an earlier attempt to show the convergence of such methods for arbitrary loss functions [17], one needed to assume that the hypothesis coefficients $\hat{\alpha}_t$ are upper bounded by a rather small constant. For this case it has been shown that the algorithm asymptotically converges to a combined hypothesis minimizing G. However, since the $\hat{\alpha}_t$'s need to be small, the algorithm requires many iterations to achieve this goal.

In [9] it has been shown that for loss functions which are (essentially) exponentially decreasing (including the loss functions of AdaBoost and Logistic regression), the loss is $\mathcal{O}(1/\sqrt{t})$ in the first $\tilde{t}$ iterations and afterwards $\mathcal{O}(\eta^{\tilde{t}-t})$. This implies linear convergence. However, this only holds, if the loss reaches zero, i.e. if the data is separable. In our work we do not need to assume separability.

An equivalent optimization problem for AdaBoost has also been considered in a paper that predates the formulation of AdaBoost [4]. This optimization problem concerns the likelihood maximization for some exponential family of distributions. In this work convergence is proven for the general non-separable case, however, only for the exponential loss, i.e. for the case of AdaBoost.[3] The framework set up in this paper is more general and we are able to treat any strictly convex loss function.

In this paper we propose a family of algorithms that are able to generate a combined hypothesis $f$ converging to the minimum of $G[f]$ (if it exists), which is a functional depending on the outputs of the function $f$ evaluated on the training set. Special cases are AdaBoost, Logistic Regression and LS-Boost. While assuming mild conditions on the base learning algorithm and the loss function G, we can show *linear convergence rates* [15] (beginning in the first iteration) of the type $G[f_{t+1}] - G[f^*] \leq \eta(G[f_t] - G[f^*])$ for some fixed $\eta \in [0, 1)$. This means that the difference to the minimum loss converges exponentially fast to zero (in the number of iterations). A similar convergence has been proven for AdaBoost in the special case of separable data [10], although the constant $\eta$ shown in [10] can be considerable smaller [see also 9]. To prove the convergence of leveraging, we exploit results of Luo & Tseng [16] for a variant of the *Gauss-Southwell method* known from numerical optimization.

Since in practice the hypothesis set $\mathcal{H}$ can be quite large, ensemble learning algorithms without any regularization often suffer from overfitting [22, 12, 2, 19]. Here, the complexity can only be controlled by the size of the base hypothesis set or by early stopping after a few iterations. However, it has been shown that *shrinkage regularization* implied by penalizing some norm of the hypothesis coefficients is the favorable strategy [6, 12]. We therefore extend our analysis to the case of $\ell_1$-norm regularized loss functions. With a slight modification this leads to a family of converging algorithms that e.g. includes the Leveraged Vector Machine [25] and a variant of LASSO [26].

In the following section we briefly review AdaBoost, Logistic Regression, and LS-Boost and cast them in a common framework. In Sec. 3 we present our main results. After re-

lating these results to leveraging algorithms, we present an extension to regularized cost functions in Sec. 4 and finally conclude.

## 2 Leveraging algorithms revisited

We first briefly review some of the most well known leveraging algorithms for classification and regression. For more details see e.g. [10, 11, 12, 8]. We work with Alg. 1 as a template for a *generic* leveraging algorithm, since these algorithms have the same algorithmical structure. Finally, we will generalize the problem and extend the notation.

**AdaBoost & Logistic Regression** are designed for classification tasks. In each iteration they call a base learning algorithm on the training set $S = \{(\mathbf{x}_1, y_1), \ldots, (\mathbf{x}_n, y_n)\} \subseteq \mathcal{X} \times \{-1, +1\}$ (cf. step 3a in Alg. 1). Here a weighting $\mathbf{d}^t = [d_1^t, \ldots, d_N^t]$ on the sample is used that is recomputed in each iteration $t$. The base learner is expected to return a hypothesis $\hat{h}_t$ from some hypothesis space[4] $\mathcal{H} := \{h_j \mid h_j : \mathcal{X} \mapsto \{-1, +1\}, j = 1, \ldots, J\}$ that has small *weighted classification error*[5] $\epsilon_t = \sum_{n=1}^{N} |d_n^t| \mathbf{I}(y_n \neq \hat{h}_t(\mathbf{x}_n))$ [10, 11], where $\mathbf{I}(\text{true}) = 1$ and $\mathbf{I}(\text{false}) = 0$. It is more convenient to work with the *edge* of $\hat{h}_t$, which is defined as $\gamma_t = 1 - 2\epsilon_t = \sum_{n=1}^{N} d_n^t \hat{h}_t(\mathbf{x}_n)$. After selecting the hypothesis, its weight $\hat{\alpha}_t$ is computed such that it minimizes a certain functional (cf. step 3b). For AdaBoost this is

$$G^{AB}(\hat{\alpha}) = \sum_{n=1}^{N} \exp\left\{-y_n \left(\hat{\alpha}\hat{h}_t(\mathbf{x}_n) + f_{t-1}(\mathbf{x}_n)\right)\right\} \tag{1}$$

and for Logistic Regression it is

$$G^{LR}(\hat{\alpha}) = \sum_{n=1}^{N} \log\left\{1 + \exp\left(-y_n (\hat{\alpha}\hat{h}_t(\mathbf{x}_n) + f_{t-1}(\mathbf{x}_n))\right)\right\}, \tag{2}$$

where $f_{t-1}$ is the combined hypothesis of the previous iteration given by $f_{t-1}(\mathbf{x}_n) = \sum_{r=1}^{t-1} \hat{\alpha}_r \hat{h}_r(\mathbf{x}_n)$. For AdaBoost it has been shown that $\hat{\alpha}_t$ minimizing (1) can be computed analytically [3]. This is true because we assumed that the hypotheses are binary valued. Similarly, for LR there exists an analytic solution of (2). The weighting $\mathbf{d}$ on the sample is updated based on the new combined hypothesis $f_t(\mathbf{x}_n) = \hat{\alpha}\hat{h}_t(\mathbf{x}_n) + f_{t-1}(\mathbf{x}_n)$: $d_n^{t+1} = y_n \exp(-y_n f_t(\mathbf{x}_n))$ and $d_n^{t+1} = y_n \frac{\exp(-y_n f_t(\mathbf{x}_n))}{1 + \exp(-y_n f_t(\mathbf{x}_n))}$, for AdaBoost and Logistic Regression, respectively.

**Least-Square-Boost** is an algorithm to solve regression tasks. In this case $S = \{(\mathbf{x}_1, y_1), \ldots, (\mathbf{x}_n, y_n)\} \subseteq \mathcal{X} \times \mathcal{Y}, \mathcal{Y} \subseteq \mathbb{R}$ and $\mathcal{H} := \{h_j \mid h_j : \mathcal{X} \mapsto \mathcal{Y}, j = 1, \ldots, J\}$. It works in a similar way as AdaBoost and LR. It first selects a hypothesis solving

$$\hat{h}_t = \operatorname*{argmin}_{\hat{h} \in \mathcal{H}} \frac{1}{2} \sum_{n=1}^{N} \left(d_n^t - \hat{h}(\mathbf{x}_n)\right)^2, \tag{3}$$

and then finds the hypothesis weight $\hat{\alpha}_t$ by minimizing the squared error of the new combined hypothesis:

$$G^{LS}(\hat{\alpha}) = \frac{1}{2} \sum_{n=1}^{N} \left(y_n - f_{t-1}(\mathbf{x}_n) - \hat{\alpha}\hat{h}_t(\mathbf{x}_n)\right)^2. \tag{4}$$

The "weighting" of the sample is computed as $d_n^{t+1} = y_n - f_t(\mathbf{x}_n)$, which is the *residual* of $f_t$ [12]. In a second version of LS-Boost, the base hypothesis and its weight are found simultaneously by solving [12]:

$$[\hat{h}_t, \hat{\alpha}_t] = \operatorname*{argmin}_{\hat{\alpha} \in \mathbb{R}, \hat{h} \in \mathcal{H}} \frac{1}{2} \sum_{n=1}^{N} \left(y_n - f_{t-1}(\mathbf{x}_n) - \hat{\alpha}\hat{h}(\mathbf{x}_n)\right)^2 \tag{5}$$

Since in (5) one reaches a lower loss function value than with (3) and (4), it might be the favorable strategy.

**Algorithm 1** – A Leveraging algorithm for the loss function G.

---

1. **Input:** $S = \langle (\mathbf{x}_1, y_1), \ldots, (\mathbf{x}_N, y_N) \rangle$, No. of Iterations $T$, Loss function $G : \mathbb{R}^N \to \mathbb{R}$
2. **Initialize:** $f_0 \equiv 0$, $d_n^1 = \text{g'}(y_n, f_0(\mathbf{x}_n))$ for all $n = 1 \ldots N$
3. **Do for** $t = 1, \ldots, T$,

   (a) Train classifier on $\{S, \mathbf{d}^t\}$ and obtain hypothesis $\hat{h}_t : \mathcal{X} \to \mathcal{Y}$

   (b) Set $\hat{\alpha}_t = \text{argmin}_{\alpha \in \mathbb{R}} G[f_t + \alpha \hat{h}_t]$

   (c) Update $f_{t+1} = f_t + \hat{\alpha}_t \hat{h}_t$ and $d_n^{t+1} = \text{g'}\left(y_n, \sum_{r=1}^{t} \hat{\alpha}_r \hat{h}_r(\mathbf{x}_n)\right)$

4. **Output:** $f_T$

---

**The General Case** These algorithms can be summarized in Alg. 1 (where case (5) is slightly degenerated, cf. Sec. 3.2) for some appropriately defined functions G and g': plug-in $G[f] = \sum_{n=1}^{N} g(y_n, f(\mathbf{x}_n))$ and choosing g as $g(y, f(\mathbf{x})) = \exp(-yf(\mathbf{x}))$ for Ada-Boost (cf. (1)), $g(y, f(\mathbf{x})) = \log(1 + \exp(-yf(\mathbf{x})))$ for Logistic Regression (cf. (2)) and $g(y, f(\mathbf{x})) = \frac{1}{2}(y - f(\mathbf{x}))^2$ for LS-Boost (cf. (4)).
It can easily be verified that the function g', used for computing the weights $\mathbf{d}$, is the derivative of g with respect to the second argument [3, 12].

**The Optimization Problem** It has been argued in [3, 18, 11, 17] and finally shown in [5] that AdaBoost and Logistic Regression under certain condition asymptotically converge to a combined hypothesis $f$ minimizing the respective loss G on the training sample, where $f$ is a linear combination of hypotheses from $\mathcal{H}$, i.e. $f_\alpha \in \text{lin}(\mathcal{H}) := \left\{ \sum_{j=1}^{J} \alpha_j h_j | h_j \in \mathcal{H}, \alpha_j \in \mathbb{R} \right\}$. Thus, they solve the optimization problem:

$$\min_{f \in \text{lin}(\mathcal{H})} G[f] = \min_{\alpha \in \mathbb{R}^J} G(H\alpha), \qquad (6)$$

where we defined a matrix $H \in \mathbb{R}^{N \times J}$ with $H_{ij} = h_j(\mathbf{x}_i)$.
To avoid confusions, note that hypotheses and coefficients generated during the iterative algorithm are marked by a hat. In the algorithms discussed so far, the optimization takes place by employing the leveraging scheme outlined in Alg. 1. The output of such an algorithm is a sequence of pairs $(\hat{\alpha}_t, \hat{h}_t)$ and a combined hypothesis $f(\mathbf{x}) = \sum_{r=1}^{t} \hat{\alpha}_r \hat{h}_r(\mathbf{x})$. With $\alpha_j^t = \sum_{r=1}^{t} \hat{\alpha}_r \mathbf{I}(\hat{h}_r = h_j)$, $j = 1, \ldots, J$, it is easy to verify that $\sum_{r}^{t} \hat{\alpha}_r \hat{h}_r(\mathbf{x}) = \sum_{j=1}^{J} \alpha_j^t h_j(\mathbf{x})$, which is in $\text{lin}(\mathcal{H})$ (note the missing hat).

**Other Preliminaries** Throughout the paper we assume the loss function $G$ is of the form

$$G[f_\alpha] = G(H\alpha) = \sum_{n=1}^{N} g(y_n, f_\alpha(\mathbf{x}_n)),$$

Although, this assumption is not necessary, the presentation becomes easier. In [7, 5, 19] a more general case of Legendre-type cost functions is considered. However, note that additive loss functions are commonly used, if one considers i.i.d.-drawn examples.

We assume that each element $H_{nj}$ and $y_n$ is finite ($j = 1, \ldots, J$, $n = 1, \ldots, N$) and $H$ does not contain a zero column. Furthermore, the function $g(y, \cdot) : \mathbb{R} \to \mathbb{R}$ is assumed to be strictly convex for all $y \in \mathcal{Y}$.

For simplicity we assume for the rest of the paper that $\mathcal{H}$ is finite and complementation closed, i.e. for every $h \in \mathcal{H}$ there exists also $-h \in \mathcal{H}$. The assumption on the finiteness is not crucial for classification (cf. footnote 4). For regression problems the hypothesis space might be infinite. This case has explicitly been analyzed in [20, 19] and goes beyond the scope of this paper (see also [27]).

## 3 Main Result

We now state a result known from the field of numerical optimization. Then we show how the reviewed leveraging algorithms fit into this optimization framework.

## 3.1 Coordinate Descent

The idea of coordinate descent is to iteratively select a coordinate, say the $j$-th, and find $\alpha_j$ such that some functional $\mathrm{F}([\alpha_1, \ldots, \alpha_j, \ldots, \alpha_T]^\top)$ is minimized with respect to $\alpha_j$. There exist several different strategies for selecting the coordinates [e.g. 15]; however, we are in particular interested in the *Gauss-Southwell*-type (GS) selection scheme: It selects the coordinate that has the largest absolute value in the gradient vector $\beta := \nabla \mathrm{F}(\alpha)$, i.e. $j = \mathrm{argmax}_{j'=1,\ldots,J} |\beta_{j'}|$. Instead of doing steps in the direction of the negative gradient as in standard *gradient descent* methods, one only changes the variable that has the largest gradient component. This can be efficient, if there are many variables and most of them are zero at the minimum.

We start with the following general convergence result, which seemed to be fallen into oblivion even in the optimization community. It will be very useful in the analysis of leveraging algorithms. Due to a lack of space we omit proofs (see [21, 19]).

**Theorem 1 (Convergence of Coordinate Descent [16]).** *Suppose* $\mathrm{G} : \mathbb{R}^N \to \mathbb{R}$ *is twice continuously differentiable and strictly convex on* $\mathrm{dom}\, \mathrm{G}$. *Assume that* $\mathrm{dom}\, \mathrm{G}$ *is open, the set of solutions* $\mathcal{S}^* \subset \mathbb{R}^J$ *to*

$$\min_{\alpha \in \mathcal{S}} \mathrm{F}(\alpha) := \mathrm{G}(H\alpha) + \gamma^\top \alpha \tag{7}$$

*is not empty, where* $H \in \mathbb{R}^{N \times J}$ *is a fixed matrix having no zero column,* $\gamma \in \mathbb{R}^J$ *fixed and* $\mathcal{S} \subseteq \mathbb{R}^J$ *is a (possibly unbounded)* box-constrained *set. Furthermore assume that the Hessian* $\nabla^2 \mathrm{G}(H\alpha^*)$ *is a positive matrix for all* $\alpha^* \in \mathcal{S}^*$. *Let* $\{\alpha^t\}$ *be the sequence generated by coordinate descent, where the coordinate selection* $j_1, j_2, \ldots$ *satisfies*

$$|\alpha_{j_t}^{t+1,j_t} - \alpha_{j_t}^t| \geq \beta \max_{j=1,\ldots,J} |\alpha_j^{t+1,j} - \alpha_j^t| \tag{8}$$

*for some* $\beta \in (0, 1]$, *where* $\alpha_j^{t+1,j}$ *is the optimal value of* $\alpha_j^{t+1}$ *if it would be selected, i.e.*

$$\alpha_j^{t+1,j} := \min_{\alpha \in \mathcal{S}_j} \mathrm{G}\left(H\alpha^t + H_j(\alpha - \alpha_j^t)\right) + \gamma_j \alpha. \tag{9}$$

*Then* $\{\alpha^t\}$ *converges to an element in* $\mathcal{S}^*$.

The coordinate selection in Thm. 1 is slightly different from the Gauss-Southwell selection rule described before. We therefore need the following:

**Proposition 2 (Convergence of GS on** $\mathbb{R}^J$**).** *Assume the conditions on* $\mathrm{G}$ *and* $H$ *as in Thm. 1. Let* $\mathcal{S}^b$ *be a convex subset of* $\mathcal{S} := \mathbb{R}^J$ *such that* $\alpha^{t,j} \in \mathcal{S}^b$. *Assume*

$$\frac{\partial^2 \mathrm{G}(H\alpha)}{\partial \alpha_j^2} \leq \eta_u \quad and \quad \frac{\partial^2 \mathrm{G}(H\alpha)}{\partial \alpha_j^2} \geq \eta_l \quad \forall \alpha \in \mathcal{S} \tag{10}$$

*holds for some fixed* $\eta_l, \eta_u > 0$. *Then a coordinate selection* $j_1, j_2, \ldots$ *satisfies* (8) *of Thm. 1, if there exists a fixed* $\delta \in (0, 1]$ *such that*

$$\left| \frac{\partial \mathrm{F}(\alpha^t)}{\partial \alpha_{j_t}^t} \right| \geq \delta \max_{j=1,\ldots,J} \left| \frac{\partial \mathrm{F}(\alpha^t)}{\partial \alpha_j^t} \right| \quad \forall t = 1, 2, \ldots \tag{11}$$

Thus the *approximate Gauss-Southwell method on* $\mathbb{R}^J$ as described above converges. To show the convergence of the second variant of LS-Boost (cf. (5)) we need the following

**Proposition 3 (Convergence of the maximal improvement scheme on** $\mathbb{R}^J$**).** *Let* $\mathrm{G}, H, \mathcal{S}$ *and* $\mathcal{S}^b$ *as in Proposition 2 and assume* (10) *holds. Then a coordinate selection* $j_1, j_2, \ldots$ *satisfies* (8), *if there exists a fixed* $\delta \in (0, 1]$ *with*

$$\mathrm{F}(\alpha^t) - \mathrm{F}(\alpha^{t+1,j_t}) \geq \delta \max_{j=1,\ldots,J} \left(\mathrm{F}(\alpha^t) - \mathrm{F}(\alpha^{t+1,j})\right) \quad \forall t = 1, 2, \ldots \tag{12}$$

Thus the *maximal improvement scheme on* $\mathbb{R}^J$ as above converges in the sense of Thm. 1. Finally we can also state a rate of convergence, which is surprisingly not worse than the rates for standard gradient descent methods:

**Theorem 4 (Rate of Convergence of Coordinate Descent, [16]).** *Assume the conditions of Thm. 1 hold. Let $\mathcal{S}^b$ as in Prop. 2 and assume (10) holds for some $\eta_l > 0$. Then we have*

$$\epsilon_{t+1} := F(\boldsymbol{\alpha}^{t+1}) - F(\boldsymbol{\alpha}^*) \leq \left(1 - \frac{1}{\eta}\right)(F(\boldsymbol{\alpha}^t) - F(\boldsymbol{\alpha}^*)), \qquad (13)$$

*where $\boldsymbol{\alpha}^t$ is the estimate after the t-th coordinate descent step, $\boldsymbol{\alpha}^*$ denotes a optimal solution, and $0 < \eta < \infty$. Especially at iteration t: $\epsilon_t \leq (1 - 1/\eta)^t \epsilon_0$.*

Following [16] one can show that the constant $\eta$ is $\mathcal{O}(\frac{\kappa^2 L J^4 N^2}{\delta^2})$, where $L$ is the Lipschitz constant of $\nabla G$ and $\kappa$ is a constant that depends on $H$ and therefore on the geometry of the hypothesis set (cf. [16, 13] for details). While the upper bound on $\eta$ can be rather large, making the convergence slow, it is important to note (i) that this is only a rough estimate of the true constant and (ii) still guarantees an exponential decrease in the error functional with the number of iterations.

### 3.2 Leveraging and Coordinate Descent

We now return from the abstract convergence results in Sec. 3.1 to our examples of leveraging algorithms, i.e. we show how to retrieve the Gauss-Southwell algorithm on $\mathbb{R}^J$ as a part of Alg. 1. For now we set $\gamma = 0$. The gradient of G with respect to $\alpha_j$ is given by

$$\frac{\partial G(\boldsymbol{\alpha})}{\partial \alpha_j} = \sum_{n=1}^N g'(y_n, f_{\boldsymbol{\alpha}}(\mathbf{x}_n)) h_j(\mathbf{x}_n) = \sum_{n=1}^N d_n h_j(\mathbf{x}_n) \qquad (14)$$

where $d_n$ is given as in step 3c of Alg. 1. Thus, the coordinate with maximal absolute gradient corresponds to the hypothesis with largest absolute edge (see definition). However, according to Proposition 2 and 3 we need to assume less on the base learner. It either has to return a hypothesis that (approximately) maximizes the edge, or alternatively (approximately) minimizes the loss function.

**Definition 5 ($\delta$-Optimality).** *A base learning algorithm L is called $\delta$-optimal, if it always returns hypotheses that either satisfy condition (11) or (12) for some fixed $\delta > 0$.*

Since we have assumed $\mathcal{H}$ is closed under complementation, there always exist two hypotheses having the same absolute gradient ($h$ and $-h$). We therefore only need to consider the hypothesis with *maximum edge* as opposed to the maximum *absolute* edge.

For classification it means: if the base learner returns the hypothesis with approximately smallest weighted training error, this condition is satisfied. It is left to show that we can apply the Thm. 1 for the loss functions reviewed in Sec. 2:

**Lemma 6.** *The loss functions of AdaBoost, Logistic regression and LS-Boost are bounded, strongly convex and fulfill the conditions in Thm. 1 on any bounded subset of $\mathbb{R}^N$.*

We can finally state the convergence result for leveraging algorithms:

**Theorem 7.** *Let G be a loss function satisfying the conditions in Thm. 1. Suppose Alg. 1 generates a sequence of hypotheses $\hat{h}_1, \hat{h}_1, \ldots$ and weights $\hat{\alpha}_1, \hat{\alpha}_2, \ldots$ using a $\delta$-optimal base learner. Assume $\{\boldsymbol{\alpha}^t\}$ with $\alpha_j^t = \sum_{r=1}^t \hat{\alpha}_r \mathbf{I}(\hat{h}_r = h_j)$ is bounded. Then any limit point of $\{\boldsymbol{\alpha}^t\}$ is a solution of (6) and converges linearly in the sense of Thm. 4.*

Note that this result in particular applies to AdaBoost, Logistic regression and the second version of LS-Boost. For the selection scheme of LS-Boost given by (3) and (4), both conditions in Definition 5 *cannot* be satisfied in general, unless $\sum_{n=1}^N h_j(\mathbf{x}_n)^2$ is constant for all hypotheses. Since $\sum_{n=1}^N (d_n - h_j(\mathbf{x}_n))^2 = \sum_{n=1}^N (h_j(\mathbf{x}_n)^2 - 2 d_n h_n(\mathbf{x}_n) + \text{const.})$, the base learner prefers hypotheses with small $\sum_{n=1}^N h_j(\mathbf{x}_n)^2$ and could therefore stop improving the objective while being not optimal (see [20, Section 4.3] and [19, Section 5] for more details).

## 4 Regularized Leveraging approaches

We have not yet exploited all features of Thm. 1. It additionally allows for box constraints and a linear function in terms of the hypothesis coefficients. Here, we are in particular interested in $\ell_1$-norm penalized loss functions of the type $F(\alpha) = G(H\alpha) + C\|\alpha\|_1$, which are frequently used in machine learning. The LASSO algorithm for regression [26] and the PBVM algorithm for classification [25] are examples. Since we assumed complementation closeness of $\mathcal{H}$, we can assume without loss of generality that a solution $\alpha^*$ satisfies $\alpha^* \geq \mathbf{0}$. We can therefore implement the $\ell_1$-norm regularization using the linear term $\gamma^\top \alpha$, where $\gamma = C\mathbf{1}$ and $C \geq 0$ is the *regularization constant*. Clearly, the regularization defines a structure of nested subsets of $\mathcal{H}$, where the hypothesis set is restricted to a smaller set for larger values of $C$.

The constraint $\alpha \geq 0$ causes some minor complications with the assumptions on the base learning algorithm. However, these can easily be resolved (cf. [21]), while not assuming more on the base learning algorithm. The first step in solving the problem is to add the additional constraint $\alpha_t \geq 0$ to the minimization with respect to $\alpha_t$ in step 3b of Alg. 1. Roughly speaking, this induces the problem that hypothesis coefficient chosen too large in a previous iteration, cannot be reduced again. To solve this problem one can check for each coefficient of a previously selected hypothesis whether *not* selecting it would violate the $\delta$-optimality condition (11) or (12). If so, the

---

**Algorithm 2** – A Leveraging algorithm for $\ell_1$-norm regularized loss $G$.

1. **Input:** Sample $S$, No. of Iterations $T$, Loss function $G : \mathbb{R}^N \to \mathbb{R}$, Reg. const. $C > 0$
2. **Initialize:** $f_0 \equiv 0$, $d_n^1 = g'(y_n, f_0(\mathbf{x}_n))$ for all $n = 1 \dots N$
3. **Do for** $t = 1, \dots, T$,
   (a) Train classifier on $\{S, \mathbf{d}^t\}$ and obtain hypothesis $\hat{h}_t : \mathcal{X} \to \mathcal{Y}$
   (b) Let $\hat{\gamma}_r = \sum_{n=1}^N d_n^t \hat{h}_r(\mathbf{x}_n)$ and $\underline{\alpha}_r = \sum_{s=1}^t \hat{\alpha}_r \mathbf{I}(\hat{h}_s = \hat{h}_r)$ for $r = 1, \dots, t$
   (c) $r^* = \operatorname{argmin}_{i \in \mathcal{J}} \hat{\gamma}_i$, where $\mathcal{J} = \{i \mid i \in \{1, \dots, t-1\}$ and $\underline{\alpha}_i > 0\}$.
   (d) **if** $\hat{\gamma}_t - C < C - \gamma_{r^*}$ **then** $\hat{h}_t = \hat{h}_{r^*}$ and $\underline{\alpha}_t = \underline{\alpha}_{r^*}$ **else** $\underline{\alpha}_t = 0$
   (e) Set $\hat{\alpha}_t = \operatorname{argmin}_{\alpha \geq -\underline{\alpha}_t} G[f_t + \alpha \hat{h}_t] + C\alpha$
   (f) Update $f_{t+1} = f_t + \hat{\alpha}_t \hat{h}_t$ and $d_n^{t+1} = g'(y_n, f_{t+1}(\mathbf{x}_n))$, $n = 1, \dots, N$
4. **Output:** $f_T$

---

algorithm selects such a coordinate for the next iteration instead of calling the base learning algorithm. This idea leads to Alg. 2 (see [21] for a detailed discussion). For this algorithm we can show the following:

**Theorem 8 (Convergence of $\ell_1$-norm penalized Leveraging).** *Assume* $G, H$ *are as Thm. 1,* $G$ *is strictly convex,* $C > 0$, *and the base learner satisfies*

$$\frac{\partial F(\alpha^t)}{\partial \alpha_{j_t}^t} \geq \delta \max_{j=1,\dots,J} \frac{\partial F(\alpha^t)}{\partial \alpha_j^t} \qquad \forall t = 1, 2, \dots \qquad (15)$$

*for* $\delta > 0$. *Then Alg. 2 converges linearly to a minimum of the regularized loss function.*

This can also be shown for a maximum-improvement like condition on the base learner, which we have to omit due to space limitation.

In [27] a similar algorithm has been suggested that solves a similar optimization problem (keeping $\|\alpha\|_1$ fixed). For this algorithm one can show order one convergence (which is weaker than linear convergence), which also holds if the hypothesis set is infinite.

## 5 Conclusion

We gave a unifying convergence analysis for a fairly general family of leveraging methods. These convergence results were obtained under rather mild assumptions on the base learner and, additionally, led to linear convergence rates. This was achieved by relating leveraging

algorithms to the Gauss-Southwell method known from numerical optimization.
While the main theorem used here was already proven in [16], its applications closes a central gap between existing algorithms and their theoretical understanding in terms of convergence. Future investigations include the generalization to infinite hypotheses spaces and an improvement of the convergence rate $\eta$. Furthermore, we conjecture that our results can be extended to many other variants of boosting type algorithms proposed recently in the literature (cf. `http://www.boosting.org`).

## Footnotes

[2]This can also be seen when analyzing a certain linear program in the dual domain (cf. [23])

[3]We will expand on this connection in the full paper (see also [14, 19]).

[4]Notice that $\mathcal{H}$ always contains only a finite number of *different* hypotheses when evaluated on the training set and is effectively finite [2].

[5]Different from common convention, we include the $y_n$ in $d_n$ to make the presentation simpler.

## References

[1] H.H. Bauschke and J.M. Borwein. Legendre functions and the method of random bregman projections. *Journal of Convex Analysis*, 4:27–67, 1997.

[2] K.P. Bennett, A. Demiriz, and J. Shawe-Taylor. A column generation algorithm for boosting. In P. Langley, editor, *Proceedings, 17th ICML*, pages 65–72. Morgan Kaufmann, 2000.

[3] L. Breiman. Prediction games and arcing algorithms. *Neural Comp.*, 11(7):1493–1518, 1999.

[4] N. Cesa-Bianchi, A. Krogh, and M. Warmuth. Bounds on approximate steepest descent for likelihood maximization in exponential families. *IEEE Trans. Inf. Th.*, 40(4):1215–1220, 1994.

[5] M. Collins, R.E. Schapire, and Y. Singer. Logistic Regression, Adaboost and Bregman distances. In *Proc. COLT*, pages 158–169, San Francisco, 2000. Morgan Kaufmann.

[6] J. Copas. Regression, prediction and shrinkage. *J.R. Statist. Soc. B*, 45:311–354, 1983.

[7] S. Della Pietra, V. Della Pietra, and J. Lafferty. Duality and auxiliary functions for bregman distances. TR CMU-CS-01-109, Carnegie Mellon University, 2001.

[8] N. Duffy and D.P. Helmbold. A geometric approach to leveraging weak learners. In P. Fischer and H. U. Simon, editors, *Proc. EuroCOLT '99*, pages 18–33, 1999.

[9] N. Duffy and D.P. Helmbold. Potential boosters? In S.A. Solla, T.K. Leen, and K.-R. Müller, editors, *NIPS*, volume 12, pages 258–264. MIT Press, 2000.

[10] Y. Freund and R.E. Schapire. A decision-theoretic generalization of on-line learning and an application to boosting. *Journal of Computer and System Sciences*, 55(1):119–139, 1997.

[11] J. Friedman, T. Hastie, and R.J. Tibshirani. Additive Logistic Regression: a statistical view of boosting. *Annals of Statistics*, 2:337–374, 2000.

[12] J.H. Friedman. Greedy function approximation. Tech. rep., Stanford University, 1999.

[13] A.J. Hoffmann. On approximate solutions of systems of linear inequalities. *Journal of Research of the National Bureau of Standards*, 49(4):263–265, October 1952.

[14] J. Kivinen and M. Warmuth. Boosting as entropy projection. In *Proc. 12th Annu. Conference on Comput. Learning Theory*, pages 134–144. ACM Press, New York, NY, 1999.

[15] D.G. Luenberger. *Linear and Nonlinear Programming*. Addison-Wesley Publishing Co., Reading, second edition, May 1984. Reprinted with corrections in May, 1989.

[16] Z.-Q. Luo and P. Tseng. On the convergence of coordinate descent method for convex differentiable minimization. *Journal of Optimization Theory and Applications*, 72(1):7–35, 1992.

[17] L. Mason, J. Baxter, P.L. Bartlett, and M. Frean. Functional gradient techniques for combining hypotheses. In *Adv. Large Margin Class.*, pages 221–247. MIT Press, 2000.

[18] T. Onoda, G. Rätsch, and K.-R. Müller. An asymptotic analysis of AdaBoost in the binary classification case. In L. Niklasson, M. Bodén, and T. Ziemke, editors, *Proc. of the Int. Conf. on Artificial Neural Networks (ICANN'98)*, pages 195–200, March 1998.

[19] G. Rätsch. *Robust Boosting via Convex Optimization*. PhD thesis, University of Potsdam, October 2001. http://mlg.anu.edu.au/˜raetsch/thesis.ps.gz.

[20] G. Rätsch, A. Demiriz, and K. Bennett. Sparse regression ensembles in infinite and finite hypothesis spaces. *Machine Learning*, 48(1-3):193–221, 2002.

[21] G. Rätsch, S. Mika, and M.K. Warmuth. On the convergence of leveraging. NeuroCOLT2 Technical Report 98, Royal Holloway College, London, 2001.

[22] G. Rätsch, T. Onoda, and K.-R. Müller. Soft margins for AdaBoost. *Machine Learning*, 42(3):287–320, March 2001. also NeuroCOLT Technical Report NC-TR-1998-021.

[23] G. Rätsch and M.K. Warmuth. Marginal boosting. NeuroCOLT2 Tech. Rep. 97, 2001.

[24] R.T. Rockafellar. *Convex Analysis*. Princeton University Press, 1970.

[25] Y. Singer. Leveraged vector machines. In S.A. Solla, T.K. Leen, and K.-R. Müller, editors, *NIPS*, volume 12, pages 610–616. MIT Press, 2000.

[26] R.J. Tibshirani. Regression selection and shrinkage via the LASSO. Technical report, Department of Statistics, University of Toronto, June 1994. ftp://utstat.toronto.edu/pub/tibs/lasso.ps.

[27] T. Zhang. A general greedy approximation algorithm with applications. In *Advances in Neural Information Processing Systems*, volume 14. MIT Press, 2002. in press.
